# THE CAPACITY OF THE KANERVA ASSOCIATIVE MEMORY IS EXPONENTIAL

P. A. Chou[1]
Stanford University, Stanford, CA 94305

## ABSTRACT

The capacity of an associative memory is defined as the maximum number of words that can be stored and retrieved reliably by an address within a given sphere of attraction. It is shown by sphere packing arguments that as the address length increases, the capacity of any associative memory is limited to an exponential growth rate of $1 - h_2(\delta)$, where $h_2(\delta)$ is the binary entropy function in bits, and $\delta$ is the radius of the sphere of attraction. This exponential growth in capacity can actually be achieved by the Kanerva associative memory, if its parameters are optimally set. Formulas for these optimal values are provided. The exponential growth in capacity for the Kanerva associative memory contrasts sharply with the sub-linear growth in capacity for the Hopfield associative memory.

## ASSOCIATIVE MEMORY AND ITS CAPACITY

Our model of an associative memory is the following. Let $(X, Y)$ be an (address, datum) pair, where $X$ is a vector of $n$ ±1s and $Y$ is a vector of $m$ ±1s, and let $(X^{(1)}, Y^{(1)}), \ldots, (X^{(M)}, Y^{(M)})$, be $M$ (address, datum) pairs stored in an associative memory. If the associative memory is presented at the input with an address $X$ that is close to some stored address $X^{(j)}$, then it should produce at the output a word $Y$ that is close to the corresponding contents $Y^{(j)}$. To be specific, let us say that an associative memory can *correct fraction $\delta$ errors* if an $X$ within Hamming distance $n\delta$ of $X^{(j)}$ retrieves $Y$ equal to $Y^{(j)}$. The Hamming sphere around each $X^{(j)}$ will be called the sphere of attraction, and $\delta$ will be called the radius of attraction.

One notion of the capacity of this associative memory is the maximum number of words that it can store while correcting fraction $\delta$ errors. Unfortunately, this notion of capacity is ill-defined, because it depends on exactly which (address, datum) pairs have been stored. Clearly, no associative memory can correct fraction $\delta$ errors for *every* sequence of stored (address, datum) pairs. Consider, for example, a sequence in which several different words are written to the same address. No memory can reliably retrieve the contents of the overwritten words. At the other extreme, any associative memory can store an unlimited number of words and retrieve them all reliably, if their contents are identical.

A useful definition of capacity must lie somewhere between these two extremes. In this paper, we are interested in the largest $M$ such that for *most* sequences of addresses $X^{(1)}, \ldots, X^{(M)}$ and *most* sequences of data $Y^{(1)}, \ldots, Y^{(M)}$, the memory can correct fraction $\delta$ errors. We define

[1]This work was supported by the National Science Foundation under NSF grant IST-8509860 and by an IBM Doctoral Fellowship.

'*most* sequences' in a probabilistic sense, as some set of sequences with total probability greater than say, .99. When all sequences are equiprobable, this reduces to the deterministic version: 99% of all sequences.

In practice it is too difficult to compute the capacity of a given associative memory with inputs of length $n$ and outputs of length $m$. Fortunately, though, it is easier to compute the asymptotic rate at which $M$ increases, as $n$ and $m$ increase, for a given family of associative memories. This is the approach taken by McEliece et al. [1] towards the capacity of the Hopfield associative memory. We take the same approach towards the capacity of the Kanerva associative memory, and towards the capacities of associative memories in general. In the next section we provide an upper bound on the rate of growth of the capacity of any associative memory fitting our general model. It is shown by sphere packing arguments that capacity is limited to an exponential rate of growth of $1 - h_2(\delta)$, where $h_2(\delta)$ is the binary entropy function in bits, and $\delta$ is the radius of attraction. In a later section it will turn out that this exponential growth in capacity can actually be achieved by the Kanerva associative memory, if its parameters are optimally set. This exponential growth in capacity for the Kanerva associative memory contrasts sharply with the sub-linear growth in capacity for the Hopfield associative memory [1].

## A UNIVERSAL UPPER BOUND ON CAPACITY

Recall that our definition of the capacity of an associative memory is the largest $M$ such that for *most* sequences of addresses $X^{(1)}, \ldots, X^{(M)}$ and *most* sequences of data $Y^{(1)}, \ldots, Y^{(M)}$, the memory can correct fraction $\delta$ errors. Clearly, an upper bound to this capacity is the largest $M$ for which there exists *some* sequence of addresses $X^{(1)}, \ldots, X^{(M)}$ such that for *most* sequences of data $Y^{(1)}, \ldots, Y^{(M)}$, the memory can correct fraction $\delta$ errors. We now derive an expression for this upper bound.

Let $\delta$ be the radius of attraction and let $D_H(X^{(j)}, d)$ be the sphere of attraction, *i.e.*, the set of all $X$s at most Hamming distance $d = \lfloor n\delta \rfloor$ from $X^{(j)}$. Since by assumption the memory corrects fraction $\delta$ errors, every address $X \in D_H(X^{(j)}, d)$ retrieves the word $Y^{(j)}$. The size of $D_H(X^{(j)}, d)$ is easily shown to be independent of $X^{(j)}$ and equal to $\nu_{n,d} = \sum_{k=0}^{d} \binom{n}{k}$, where $\binom{n}{k}$ is the binomial coefficient $n!/k!(n-k)!$. Thus out of a total of $2^n$ $n$-bit addresses, at least $\nu_{n,d}$ addresses retrieve $Y^{(1)}$, at least $\nu_{n,d}$ addresses retrieve $Y^{(2)}$, at least $\nu_{n,d}$ addresses retrieve $Y^{(3)}$, and so forth. It follows that the total number of distinct $Y^{(j)}$s can be at most $2^n/\nu_{n,d}$. Now, from Stirling's formula it can be shown that if $d \leq n/2$, then $\nu_{n,d} = 2^{nh_2(d/n)+\mathcal{O}(\log n)}$, where $h_2(\delta) = -\delta \log_2 \delta - (1-\delta)\log_2(1-\delta)$ is the binary entropy function in bits, and $\mathcal{O}(\log n)$ is some function whose magnitude grows more slowly than a constant times $\log n$. Thus the total number of distinct $Y^{(j)}$s can be at most $2^{n(1-h_2(\delta))+\mathcal{O}(\log n)}$. Since any set containing '*most* sequences' of $M$ $m$-bit words will contain a large number of distinct words (if $m$ is

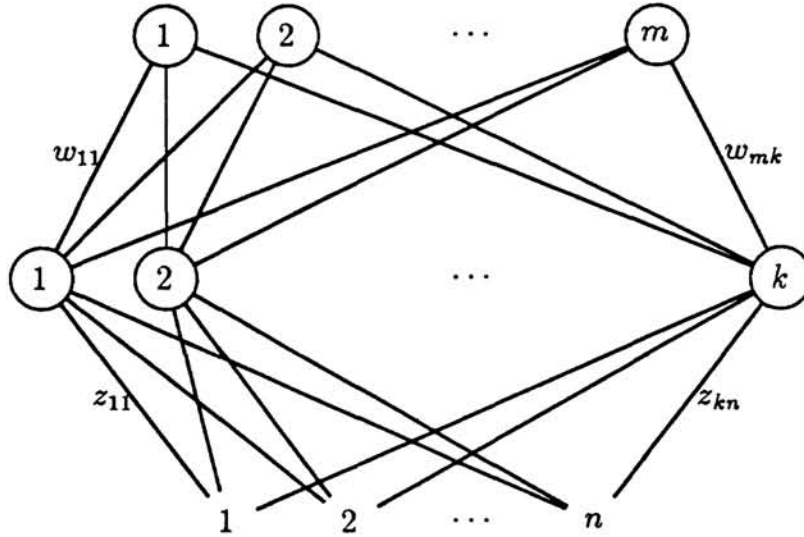

Figure 1: Neural net representation of the Kanerva associative memory. Signals propagate from the bottom (input) to the top (output). Each arc multiplies the signal by its weight; each node adds the incoming signals and then thresholds.

sufficiently large --- see [2] for details), it follows that

$$M \leq 2^{n(1-h_2(\delta))+\mathcal{O}(\log n)}. \tag{1}$$

In general a function $f(n)$ is said to be $\mathcal{O}(g(n))$ if $f(n)/g(n)$ is bounded, *i.e.*, if there exists a constant $\alpha$ such that $|f(n)| \leq \alpha|g(n)|$ for all $n$. Thus (1) says that there exists a constant $\alpha$ such that $M \leq 2^{n(1-h_2(\delta))+\alpha\log n}$. It should be emphasized that since $\alpha$ is unknown, this bound has no meaning for fixed $n$. However, it indicates that asymptotically in $n$, the maximum exponential rate of growth of $M$ is $1 - h_2(\delta)$.

Intuitively, only a sequence of addresses $X^{(1)}, \ldots, X^{(M)}$ that optimally pack the address space $\{-1, +1\}^n$ can hope to achieve this upper bound. Remarkably, *most* such sequences are optimal in this sense, when $n$ is large. The Kanerva associative memory can take advantage of this fact.

## THE KANERVA ASSOCIATIVE MEMORY

The Kanerva associative memory [3,4] can be regarded as a two-layer neural network, as shown in Figure 1, where the first layer is a preprocessor and the second layer is the usual Hopfield style array. The preprocessor essentially encodes each $n$-bit input address into a very large $k$-bit internal representation, $k \gg n$, whose size will be permitted to grow exponentially in $n$. It does not seem surprising, then, that the capacity of the Kanerva associative memory can grow exponentially in $n$, for it is known that the capacity of the Hopfield array grows almost linearly in $k$, assuming the coordinates of the $k$-vector are drawn at random by independent flips of a fair coin [1].

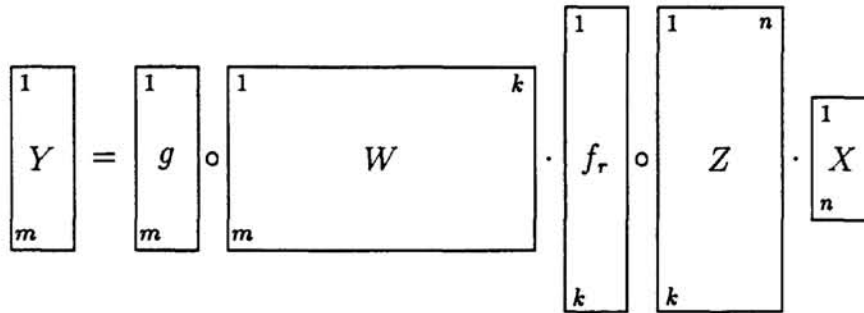

Figure 2: Matrix representation of the Kanerva associative memory. Signals propagate from the right (input) to the left (output). Dimensions are shown in the box corners. Circles stand for functional composition; dots stand for matrix multiplication.

In this situation, however, such an assumption is ridiculous: Since the $k$-bit internal representation is a function of the $n$-bit input address, it can contain at most $n$ bits of information, whereas independent flips of a fair coin contain $k$ bits of information. Kanerva's primary contribution is therefore the specification of the preprocessor, that is, the specification of how to map each $n$-bit input address into a very large $k$-bit internal representation.

The operation of the preprocessor is easily described. Consider the matrix representation shown in Figure 2. The matrix $Z$ is randomly populated with $\pm 1$s. This randomness assumption is required to ease the analysis. The function $f_r$ is 1 in the $i$th coordinate if the $i$th row of $Z$ is within Hamming distance $r$ of $X$, and is 0 otherwise. This is accomplished by thresholding the $i$th input against $n - 2r$. The parameters $r$ and $k$ are two essential parameters in the Kanerva associative memory. If $r$ and $k$ are set correctly, then the number of 1s in the representation $f_r(ZX)$ will be very small in comparison to the number of 0s. Hence $f_r(ZX)$ can be considered to be a sparse internal representation of $X$.

The second stage of the memory operates in the usual way, except on the internal representation of $X$. That is, $Y = g(Wf_r(ZX))$, where

$$W = \sum_{j=1}^{M} Y^{(j)}[f_r(ZX^{(j)})]^t, \tag{2}$$

and $g$ is the threshold function whose $i$th coordinate is $+1$ if the $i$th input is greater than 0 and $-1$ is the $i$th input is less than 0. The $i$th column of $W$ can be regarded as a memory location whose address is the $i$th row of $Z$. Every $X$ within Hamming distance $r$ of the $i$th row of $Z$ accesses this location. Hence $r$ is known as the *access radius*, and $k$ is the *number of memory locations*.

The approach taken in this paper is to fix the linear rate $\rho$ at which $r$ grows with $n$, and to fix the exponential rate $\kappa$ at which $k$ grows with $n$. It turns out that the capacity then grows at a fixed exponential rate $C_{\rho,\kappa}(\delta)$, depending on $\rho$, $\kappa$, and $\delta$. These exponential rates are sufficient to overcome the standard loose but simple polynomial bounds on the errors due to combinatorial approximations.

## THE CAPACITY OF THE KANERVA ASSOCIATIVE MEMORY

Fix $0 \leq \kappa \leq 1$, $0 \leq \rho \leq 1/2$, and $0 \leq \delta \leq \min\{2\rho, 1/2\}$. Let $n$ be the input address length, and let $m$ be the output word length. It is assumed that $m$ is at most polynomial in $n$, i.e., $m = \exp\{\mathcal{O}(\log n)\}$. Let $r = \lfloor \rho n \rfloor$ be the access radius, let $k = 2^{\lfloor \kappa n \rfloor}$ be the number of memory locations, and let $d = \lfloor \delta n \rfloor$ be the radius of attraction. Let $M_n$ be the number of stored words. The components of the $n$-vectors $X^{(1)}, \ldots, X^{(M_n)}$, the $m$-vectors $Y^{(1)}, \ldots, Y^{(M_n)}$, and the $k \times n$ matrix $Z$ are assumed to be IID equiprobable $\pm 1$ random variables. Finally, given an $n$-vector $X$, let $Y = g(W f_r(ZX))$ where $W = \sum_{j=1}^{M_n} Y^{(j)} [f_r(ZX^{(j)})]^t$.

Define the quantity

$$ C_{\rho,\kappa}(\delta) = \begin{cases} 2\delta + 2(1-\delta)h(\frac{\rho - \delta/2}{1-\delta}) + \kappa - 2h(\rho) & \text{if } \kappa \leq \kappa_0(\rho) \\ C_{\rho,\kappa_0(\rho)}(\delta) & \text{if } \kappa > \kappa_0(\rho) \end{cases}, \quad (3) $$

where

$$ \kappa_0(\rho) = 2h(\rho) - 2\gamma - 2(1-\gamma)h(\frac{\rho - \gamma/2}{1-\gamma}) + 1 - h(\gamma) \quad (4) $$

and

$$ \gamma = \frac{3}{4} - \sqrt{\frac{9}{16} - 2\rho(1-\rho)}. $$

*Theorem:* If

$$ M_n \leq 2^{nC_{\rho,\kappa}(\delta) + \mathcal{O}(\log n)} $$

then for all $\epsilon > 0$, all sufficiently large $n$, all $j \in \{1, \ldots, M_n\}$, and all $X \in D_H(X^{(j)}, d)$,

$$ P\{Y \neq Y^{(j)}\} < \epsilon. $$

*Proof:* See [2].

*Interpretation:* If the exponential growth rate of the number of stored words $M_n$ is asymptotically less than $C_{\rho,\kappa}(\delta)$, then for every sufficiently large address length $n$, there is some realization of the $n \times 2^{n\kappa}$ preprocessor matrix $Z$ such that the associative memory can correct fraction $\delta$ errors for *most* sequences of $M_n$ (address, datum) pairs. Thus $C_{\rho,\kappa}(\delta)$ is a lower bound on the exponential growth rate of the capacity of the Kanerva associative memory with access radius $n\rho$ and number of memory locations $2^{n\kappa}$.

Figure 3 shows $C_{\rho,\kappa}(\delta)$ as a function of the radius of attraction $\delta$, for $\kappa = \kappa_0(\rho)$ and $\rho = 0.1$, 0.2, 0.3, 0.4 and 0.45. For any fixed access radius $\rho$, $C_{\rho,\kappa_0(\rho)}(\delta)$ decreases as $\delta$ increases. This reflects the fact that fewer (address, datum) pairs can be stored if a greater fraction of errors must be corrected. As $\rho$ increases, $C_{\rho,\kappa_0(\rho)}(\delta)$ begins at a lower point but falls off less steeply. In a moment we shall see that $\rho$ can be adjusted to provide the optimal performance for a given $\delta$.

Not shown in Figure 3 is the behavior of $C_{\rho,\kappa}(\delta)$ as a function of $\kappa$. However, the behavior is simple. For $\kappa > \kappa_0(\rho)$, $C_{\rho,\kappa}(\delta)$ remains unchanged, while for $\kappa \leq \kappa_0(\rho)$, $C_{\rho,\kappa}(\delta)$ is simply shifted down by the difference $\kappa_0(\rho) - \kappa$. This establishes the conditions under which the Kanerva associative memory is robust against random component failures. Although increasing the number of memory locations beyond $2^{n\kappa_0(\rho)}$ does not increase the capacity, it does increase robustness. Random

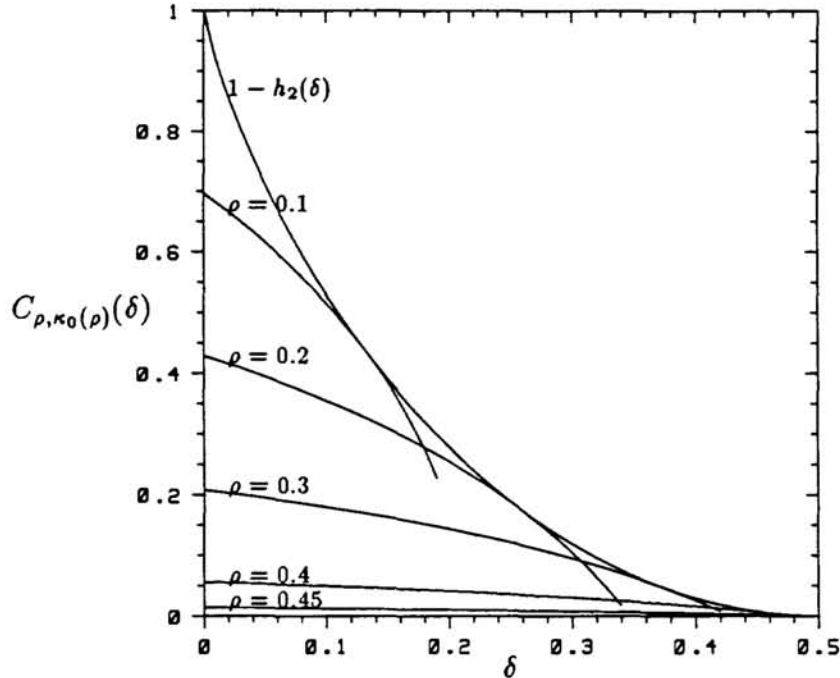

Figure 3: Graphs of $C_{\rho,\kappa_0(\rho)}(\delta)$ as defined by (3). The upper envelope is $1 - h_2(\delta)$.

component failures will not affect the capacity until so many components have failed that the number of surviving memory locations is less than $2^{n\kappa_0(\rho)}$.

Perhaps the most important curve exhibited in Figure 3 is the sphere packing upper bound $1 - h_2(\delta)$, which is achieved for a particular $\rho$ by $\delta = \frac{3}{4} - \sqrt{\frac{9}{16} - 2\rho(1-\rho)}$. Equivalently, the upper bound is achieved for a particular $\delta$ by $\rho$ equal to

$$\rho_0(\delta) = \tfrac{1}{2} - \sqrt{\tfrac{1}{4} - \tfrac{3}{4}\delta(1 - \tfrac{2}{3}\delta)}. \tag{5}$$

Thus (4) and (5) specify the optimal values of the parameters $\kappa$ and $\rho$, respectively. These functions are shown in Figure 4. With these optimal values, (3) simplifies to

$$C_{\rho,\kappa}(\delta) = 1 - h(\delta),$$

the sphere packing bound.

It can also be seen that for $\delta = 0$ in (3), the exponential growth rate of the capacity is asymptotically equal to $\kappa$, which is the exponential growth rate of the number of memory locations, $k_n$. That is, $M_n = 2^{n\kappa + \mathcal{O}(\log n)} = k_n \cdot 2^{\mathcal{O}(\log n)}$. Kanerva [3] and Keeler [5] have argued that the capacity at $\delta = 0$ is proportional to the number of memory locations, i.e., $M_n = k_n \cdot \beta$, for some constant $\beta$. Thus our results are consistent with those of Kanerva and Keeler, provided the 'polynomial' $2^{\mathcal{O}(\log n)}$ can be proved to be a constant. However, the usual statement of their result, $M = k \cdot \beta$, that the capacity is simply proportional to the number of memory locations, is false, since in light of the universal

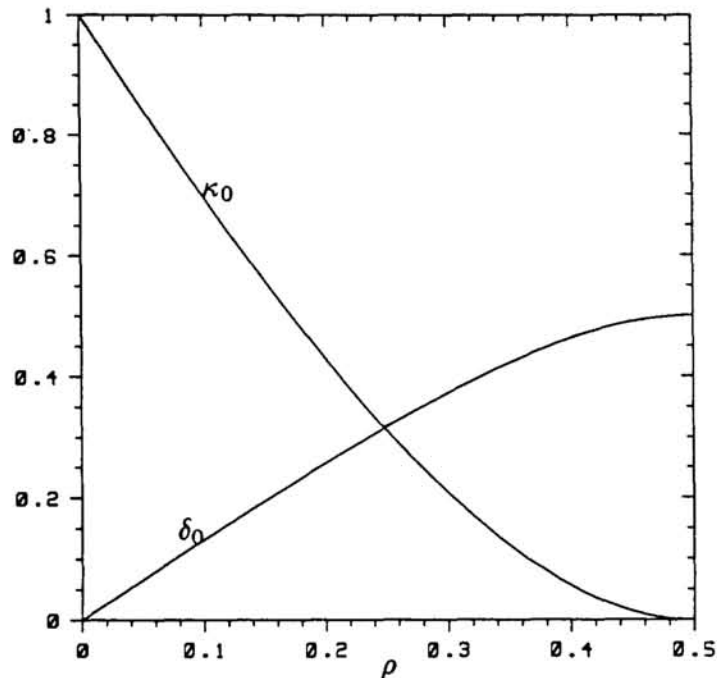

Figure 4: Graphs of $\kappa_0(\rho)$ and $\delta_0(\rho)$, the inverse of $\rho_0(\delta)$, as defined by (4) and (5).

upper bound, it is impossible for the capacity to grow without bound, with no dependence on the dimension $n$. In our formulation, this difficulty does not arise because we have explicitly related the number of memory locations to the input dimension: $k_n = 2^{n\kappa}$. In fact, our formulation provides explicit, coherent relationships between all of the following variables: the capacity $M$, the number of memory locations $k$, the input and output dimensions $n$ and $m$, the radius of attraction $\delta$, and the access radius $\rho$. We are therefore able to generalize the results of [3,5] to the case $\delta > 0$, and provide explicit expressions for the asymptotically optimal values of $\rho$ and $\kappa$ as well.

## CONCLUSION

We described a fairly general model of associative memory and selected a useful definition of its capacity. A universal upper bound on the growth of the capacity of such an associative memory was shown by a sphere packing argument to be exponential with rate $1 - h_2(\delta)$, where $h_2(\delta)$ is the binary entropy function and $\delta$ is the radius of attraction. We reviewed the operation of the Kanerva associative memory, and stated a lower bound on the exponential growth rate of its capacity. This lower bound meets the universal upper bound for optimal values of the memory parameters $\rho$ and $\kappa$. We provided explicit formulas for these optimal values. Previous results for $\delta = 0$ stating that the capacity of the Kanerva associative memory is proportional to the number of memory locations cannot be strictly true. Our formulation corrects the problem and generalizes those results to the case $\delta > 0$.

REFERENCES

1. R.J. McEliece, E.C. Posner, E.R. Rodemich, and S.S. Venkatesh, ''The capacity of the Hopfield associative memory,'' *IEEE Transactions on Information Theory*, submitted.
2. P.A. Chou, ''The capacity of the Kanerva associative memory,'' *IEEE Transactions on Information Theory*, submitted.
3. P. Kanerva, ''Self-propagating search: a unified theory of memory,'' Tech. Rep. CSLI-84-7, Stanford Center for the Study of Language and Information, Stanford, CA, March 1984.
4. P. Kanerva, ''Parallel structures in human and computer memory,'' in *Neural Networks for Computing*, (J.S. Denker, ed.), New York: American Institute of Physics, 1986.
5. J.D. Keeler, ''Comparison between sparsely distributed memory and Hopfield-type neural network models,'' Tech. Rep. RIACS TR 86.31, NASA Research Institute for Advanced Computer Science, Mountain View, CA, Dec. 1986.